# Correlation Codes in Neuronal Populations

**Maoz Shamir and Haim Sompolinsky**
Racah Institute of Physics and Center for Neural Computation,
The Hebrew University of Jerusalem, Jerusalem 91904, Israel
$\{maoz, haim @ fiz.huji.ac.il\}$

## Abstract

Population codes often rely on the tuning of the mean responses to the stimulus parameters. However, this information can be greatly suppressed by long range correlations. Here we study the efficiency of coding information in the second order statistics of the population responses. We show that the Fisher Information of this system grows linearly with the size of the system. We propose a bilinear readout model for extracting information from correlation codes, and evaluate its performance in discrimination and estimation tasks. It is shown that the main source of information in this system is the stimulus dependence of the variances of the single neuron responses.

## 1   Introduction

Experiments in the last years have shown that in many cortical areas, the fluctuations in the responses of neurons to external stimuli are significantly correlated [1, 2, 3, 4], raising important questions regarding the computational implications of neuronal correlations. Recent theoretical studies have addressed the issue of how neuronal correlations affect the efficiency of population coding [4, 5, 6]. It is often assumed that the information about stimuli is coded mainly in the mean neuronal responses, *e.g.*, in the tuning of the mean firing rates, and that by averaging the tuned responses across large populations, an accurate estimate can be obtained despite the significant noise in the single neuron responses. Indeed, for uncorrelated neurons the Fisher Information of the population is extensive [7]; namely, it increases linearly with the number of neurons in the population. Furthermore, it has been shown that this extensive information can be extracted by relatively simple linear readout mechanisms [7, 8]. However, it was recently shown [6] that positive correlations which vary smoothly with space may drastically suppress the information in the mean responses. In particular, the Fisher Information of the system saturates to a finite value as the system size grows. This raises questions about the computational utility of neuronal population codes.

Neuronal population responses can represent information in the higher order statistics of the responses [3], not only in their means. In this work, we study the accuracy of coding information in the second order statistics. We call such schemes correlation codes. Specifically, we assume that the neuronal responses obey multivariate Gaussian statistics governed by a stimulus-dependent correlation matrix. We ask whether the Fisher Information of such a system is extensive even in the presence of strong correlations in the neuronal

noise. Secondly, we inquire how information in the second order statistics can be efficiently extracted.

## 2 Fisher Information of a Correlation Code

Our model consists of a system of $N$ neurons that code a 2D angle $\theta$, $0 \leq \theta \leq 2\pi$. Their stochastic response is given by a vector of activities $\{r_i\}_{i=1}^N$ where $r_i$ is the activity of the $i$-th neuron in the presence of a stimulus $\theta$, and is distributed according to a multivariate Gaussian distribution

$$P(\mathbf{r} \mid \theta) = \frac{1}{Z} \exp \left\{ -\frac{1}{2} (\mathbf{r} - \mathbf{f}(\theta))^t \mathbf{C}(\theta)^{-1} (\mathbf{r} - \mathbf{f}(\theta)) \right\} \tag{1}$$

Here $f_i(\theta)$ is the mean activity of the $i$-th neuron and its dependence on $\theta$ is usually referred to as the tuning curve of the neuron; $\mathbf{C}(\theta)$ is the correlation matrix; and $Z$ is a normalization constant. Here we shall limit ourselves to the case of multiplicative modulation of the correlations. Specifically we use

$$C_{ij}(\theta) = b_i(\theta) b_j(\theta) \bar{C}_{ij} \tag{2}$$

$$\bar{C}_{ij} = \bar{C}(\phi_i - \phi_j) = \delta_{ij} + c(1 - \delta_{ij}) \exp \left( -\frac{|\phi_i - \phi_j|}{\rho} \right) \tag{3}$$

$$b_i(\theta) = b(\phi_i - \theta) = \exp \left( \frac{\cos(\phi_i - \theta)}{\sigma^2} \right) \tag{4}$$

where $c$ and $\rho$ are the correlation strength and correlation length respectively; $\sigma$ defines the tuning width of the correlations; and $\phi_i$ denotes the angle at which the variance of the $i$-th neuron, $b_i^2(\theta)$, is maximal. An example is shown in Fig. 1. It is important to note that the variance adds a contribution to $C_{ij}$ which is larger than the contribution of the smooth part of the correlations. For reasons that will become clear below, we write,

$$C_{ij}(\theta) = C_{ij}^s(\theta) + C_i^d(\theta) \delta_{ij} \tag{5}$$

where $C_{ij}^s$ denotes the smooth part of the correlation matrix and $C_i^d$ the discontinuous diagonal part, which in the example of Eqs. (2)-(4) is

$$C_i^d(\theta) = (1 - c) b_i^2(\theta). \tag{6}$$

A useful measure of the accuracy of a population code is the Fisher Information (FI). In the case of uncorrelated populations it is well known that FI increases linearly with system size [7], indicating that the accuracy of the population coding improves as the system size is increased. Furthermore, it has been shown that relatively simple, linear schemes can provide reasonable readout models for extracting the information in uncorrelated populations [8]. In the case of a correlated multivariate Gaussian distribution, FI is given as $J = J_{mean} + J_{corr}$, where

$$J_{mean} = \mathbf{f}(\theta)'^t \mathbf{C}(\theta)^{-1} \mathbf{f}(\theta)' \tag{7}$$

$$J_{corr} = \frac{1}{2} Tr[\mathbf{C}(\theta)^{-1}(\mathbf{C}(\theta))']^2 \tag{8}$$

where $\mathbf{f}'$ and $\mathbf{C}'$ denote derivatives of $\mathbf{f}$ and $\mathbf{C}$ with respect to $\theta$, respectively. The form of these terms reveals that in general the correlations play two roles. First they control the efficiency of the information encoded in the mean activities $\mathbf{f}(\theta)$ (note the dependence of $J_{mean}$ on $C$). Secondly, $\mathbf{C}(\theta)$ provides an additional source of information about the stimulus ($J_{corr}$). When the correlations are independent of the stimulus, i.e. $b_i(\theta) = const$, it was shown [6] that positive correlations, $c > 0$, with long correlation length, $\rho = \mathcal{O}(1)$,

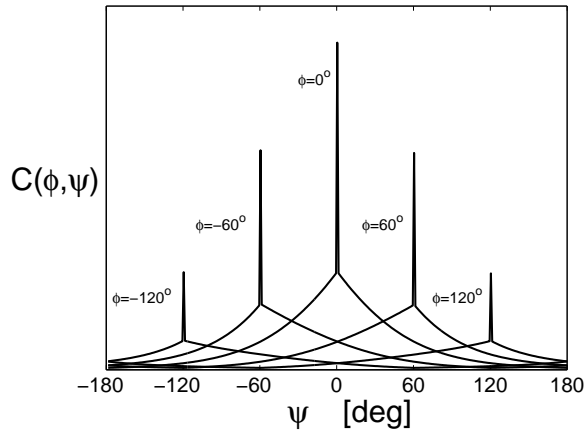

Figure 1: The stimulus-dependent correlation matrix, Eqs. (2)-(4), depicted as a function of two angles, $C(\phi, \psi)$, where $\phi = \phi_i - \theta$ and $\psi = \phi_i - \theta$. Here, $c = 0.3$, $\rho = 1$ and $\sigma = \pi/2$.

cause the saturation of FI to a finite limit at large $N$. This implies that in the presence of such correlations, population averaging cannot overcome the noise even in large networks. This analysis however, [6], did not take into account stimulus-dependent correlations, which is the topic of the present work.

Analyzing the $N$ dependence of $J_{corr}$, Eq. (8), we find it useful to write

$$J_{corr} = J_d + J_s, \qquad (9)$$

where

$$J_d = \frac{1}{2} \sum_{i=1}^{N} \left( \frac{(C_i^d(\theta))'}{C_i^d(\theta)} \right)^2 \qquad (10)$$

is FI of an uncorrelated population with stimulus-dependent variance which equals $C_i^d$, and scales linearly with $N$; $J_s = J_{corr} - J_d$. Evaluating these terms for the multiplicative model, Eq. (2), we find that $J_s$ is positive, so that $J \geq J_d$. Furthermore, numerical evaluation of this term shows that $J_s$ saturates at large $N$ to a small finite value, so that for large $N$

$$J_{corr} \approx J_d = 2N \int_0^{2\pi} \frac{d\phi}{2\pi} \left( \frac{b'(\phi)}{b(\phi)} \right)^2 \qquad (11)$$

as shown in Fig. 2. We thus conclude that $J_{corr}$ increases linearly with $N$ and is equal, for large $N$, to the FI of *variance coding* namely to $J$ of an independent population in which information is encoded in their activity variances.

Since in our system the information is encoded in the second order statistics of the population responses, it is obvious that linear readouts are inadequate. This raises the question of whether there are relatively simple *nonlinear* readout models for such systems. In the next sections we will study bilinear readouts and show that they are useful models for extracting information from correlation codes.

## 3  A Bilinear Readout for Discrimination Tasks

In a two-interval discrimination task the system is given two sets of neuronal activities $\mathbf{r}^{(1)}, \mathbf{r}^{(2)}$ generated by two proximal stimuli $\theta$ and $\theta + \delta\theta$ and must infer which stimulus generated which activity. The Maximum-Likelihood (ML) discrimination yields the

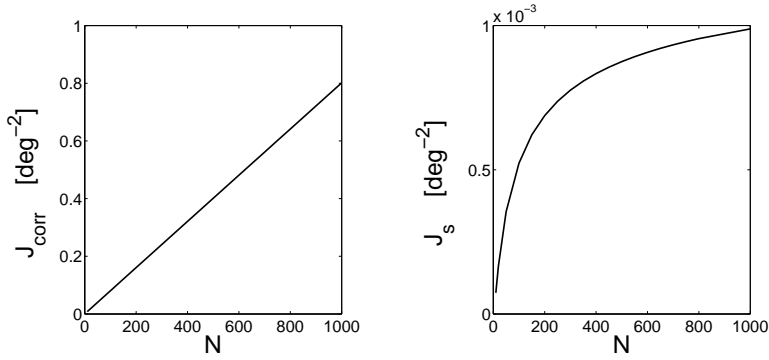

Figure 2: (a) Fisher Information, $J_{corr}$, of the stimulus-dependent correlations, Eqs. (2)-(4), as a function of the number of neurons in the system. In (b) we show the difference between the full FI and the contribution of the diagonal term, $J_s$ - as defined by Eq. (9). Here $c = 0.3$, $\rho = 1$ and $\sigma = \pi/4$. Note the different scales in (a) and (b).

probability of error given by $H(d'/\sqrt{2})$, where $H(x) = (2\pi)^{-1/2} \int_x^\infty dx e^{-x^2/2}$ and the discriminability $d'$ equals

$$d' = |\delta\theta| \sqrt{J(\theta)}. \tag{12}$$

It has been previously shown that in the case of uncorrelated populations with mean coding, the optimal linear readouts achieves the Maximum-Likelihood discrimination performance in large N [7].

In order to isolate the properties of correlation coding we will assume that no information is coded in the average firing rates of the neurons, and take $\mathbf{f} = \mathbf{0}$ hereafter. We suggest a bilinear readout as a simple generalization of the linear readout to correlation codes. In a discrimination task the bilinear readout makes a decision according to the sign of

$$q = \sum_{ij} W_{ij} \left( r_i^{(1)} r_j^{(1)} - r_i^{(2)} r_j^{(2)} \right) \tag{13}$$

where a $+(-)$ decision refers to $\theta(\theta + \delta\theta)$. Maximizing the signal-to-noise ratio of this rule, the optimal bilinear discriminator (OBD) matrix is given by

$$W_{ij} = (C^{-1}(\theta)'_{ij}). \tag{14}$$

Using the optimal weights to evaluate the discrimination error we obtain that in large $N$ the performance of the OBD saturates the ML performance, Eq. (12). Thus, since FI of this model increases linearly with the size of the system, the discriminability increases as $\sqrt{N}$.

Since the correlation matrix $\mathbf{C}$ depends on the stimulus, $\theta$, the OBD matrix, Eq. (14), will also be stimulus dependent. Thus, although the OBD is *locally* efficient, it cannot be used as such as a *global* efficient readout.

## 4   A Bilinear Readout for Angle Estimation

### 4.1   Optimal bilinear readout for estimation

To study the global performance of bilinear readouts we investigate bilinear readouts which minimize the square error of estimating the angle averaged over the whole range of $\theta$. For convenience we use complex notation for the encoded angle, and write $\hat{z}$ as the estimator

of $z = e^{i\theta}$. Let

$$\hat{z} = \sum_{ij} W_{ij} r_i r_j \tag{15}$$

where $W_{ij}$ are stimulus independent complex weights. We define the optimal bilinear estimator (OBE) as the set of weights $\mathbf{W}$ that minimizes on average the quadratic estimation error of an unbiased estimator. This error is given by

$$E(\mathbf{W}) = \frac{1}{2} \int \frac{d\theta}{2\pi} \langle |\delta \hat{z}|^2 \rangle - \int \frac{d\theta}{2\pi} \lambda(\theta) \langle \hat{z} \rangle \tag{16}$$

where $\lambda(\theta)$ is the Lagrange multiplier of the constraint $\langle \hat{z} \{\mathbf{r}(\theta)\} \rangle = z(\theta)$. In general, it is impossible to find a perfectly unbiased estimator for a continuously varied stimulus, using a finite number of weights. However, in the case of angle estimation, we can employ the underlying rotational symmetry to generate such an estimator. For this we use the symmetry of the correlation matrix, Eq. (2). In this case one can show that the Lagrange multipliers have the simple form of $\lambda(\theta) = \lambda e^{i\theta}$, and the OBE weight matrix is in the form of

$$W_{ij} = w(\phi_i - \phi_j) \exp\left(i \frac{\phi_i + \phi_j}{2}\right) \tag{17}$$

where $w(\phi) = w(-\phi)$ and $w(\phi) = -w(\phi + 2\pi)$ . This form of a readout matrix, Eq. (17), guarantees that the estimator will be unbiased. Using these symmetry properties, $w(\phi)$ can be written in the following form (for even $N$)

$$w(\phi) \quad \propto \quad \delta_{\phi,0} + \sum_{n=1}^{N/2-1} w^{(n)} \cos[(n - \frac{1}{2})\phi] , \quad -\pi < \phi \leq \pi \tag{18}$$

Figure 3 (a) presents an example of the function $w(\phi)$. These numerical results (Fig. 3 (a)) also suggest that the function $w(\phi)$ is mainly determined by a few harmonics plus a delta peak at $\phi = 0$. Below we will use this fact to study simpler forms of bilinear readout.

Further analysis of the OBE performance in the large $N$ limit yields the following asymptotic result

$$\langle (\delta \hat{\theta})^2 \rangle^{-1} = N \frac{\left| \int \frac{d\phi}{2\pi} C_i^d(\phi) e^{i\phi} \right|^2}{\int \frac{d\phi}{2\pi} (C_i^d(\phi))^2 (1 - \cos(2\phi))} \tag{19}$$

Figure 3 (b) shows the numerical calculation of the OBE error (open circles) as a function of $N$. The dashed line is the asymptotic behavior, given by Eq. (19). The dotted line is the Creamer-Rao bound. From the graph one can see that the estimation efficiency of this readout grows linearly with the size of the system, $N$, but is lower than the bound.

## 4.2   Truncated bilinear readout

Motivated by the simple structure of the optimal readout matrix observed in Fig. 3 (a), we studied a bilinear readout of the form of Eqs. (17) and (18) with $w(\phi)$ which has a delta function peak at the origin plus a few harmonics. Restricting the number of harmonics to relatively small integers, we evaluated numerically the optimal values of the coefficients $w^{(n)}$ for large systems. Surprisingly we found that for small $p$ and large $N$, these coefficients approach a value which is independent of the specifics of the model and equals $w^{(n)} = -2/N$, yielding a bilinear weight matrix of the form

$$W_{ij} \propto \left( \delta_{ij} - \frac{2}{N} \sum_{n=1}^{p} \cos[(n - \frac{1}{2})(\phi_i - \phi_j)] \right) \exp\left(i \frac{\phi_i + \phi_j}{2}\right) . \tag{20}$$

Figure 4 shows the numerical results for the squared average error of this readout for several values of $p \leq 6$ and $N \leq 2000$. The results of Fig. 4 show that for a given $p$ the

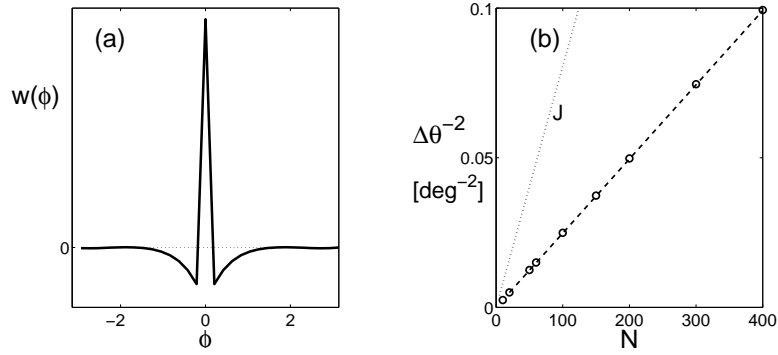

Figure 3: (a) Profile of $w(\phi)$, Eq. (17), for the OBE with $N = 30$. (b) Numerical evaluation of one over the squared estimation error, for the optimal bilinear readout in the multiplicative modulation model (open circles). The dashed line is the asymptotic behavior, given by Eq. (19). Here $\Delta\theta = (\langle(\delta\hat{\theta})^2\rangle)^{1/2}$, for the optimal bilinear readout in the multiplicative modulation model. The dotted line is the FI bound. In these simulations $c = 0.3$, $\rho = 1$ and $\sigma = \pi/4$ were used.

inverse square error initially increases linearly with $N$ but saturates in the limit of large $N$. However, the saturation size $N_{sat}(p)$ increases rapidly with $p$. The precise form of $N_{sat}(p)$ depends on the specifics of the correlation model. For the exponentially decaying correlations assumed in Eq. (2), we find $N_{sat} \propto p^3$. Figure 4 shows that for this range of $N$, and $p = 6$ the deviations of the inverse square error from linearity are small. Thus, in the regime $1 << N << N_{sat}(p)$, $\langle(\delta\hat{\theta})^2\rangle$ is given by the asymptotic behavior, Eq. (19), shown by the dashed line.

We thus conclude that the OBE (with unlimited $p$) will generate an inverse square estimation error which increases linearly with $N$ with a coefficient given by Eq. (19), and that this value can be achieved for reasonable values of $N$ by an approximate bilinear weight matrix, of the form of Eq. (20), with small $p$. The asymptotic result, Eq. (19), is smaller than the optimal value given by the full FI, Eq. (11), see Fig. 4 (dotted line). In fact, it is equal to the error of an *independent* population with a variance which equals $C_i^d(\theta)$ and a quadratic population vector readout of the form

$$\hat{z} \propto \sum_{i=1}^{N} r_i^2 e^{i\phi_i} \tag{21}$$

It is important to note that in the presence of correlations, the quadratic readout of Eq. (21) is very inefficient, yielding a finite error for large $N$ as shown in Fig. 4 (line marked 'quadratic').

## 5 Discussion

To understand the reason for the simple form of the approximately optimal bilinear weight matrix, Eq. (20), we rewrite Eq. (15) with $\mathbf{W}$ of Eq. (20) as

$$\hat{z} = \sum_{i=1}^{N} r_i \hat{r}_i e^{i\phi_i} \tag{22}$$

$$\hat{r}_i = \sum_{j=1}^{N} \left( \delta_{ij} - \frac{1}{N} \sum_{n=-p}^{p} e^{in(\phi_i - \phi_j)} \right) r_j \tag{23}$$

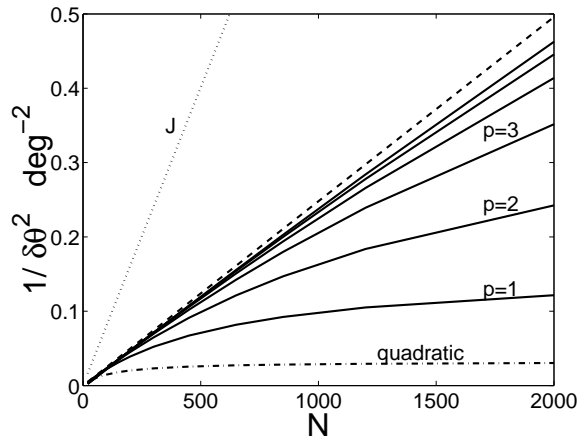

Figure 4: Inverse square estimation error of the finite-$p$ approximation for the OBE, Eq. (20). Solid curves from the bottom $p = 1, 2 \ldots 6$. The bottom curve is $p = 0$. The dashed line is the asymptotic behavior, given by Eq. (19). The FI bound is shown by the dotted line. For the simulations $c = 0.3$, $\rho = 1$ and $\sigma = \pi/4$ were used.

Comparing this form with Eq. (21) it can be seen that our readout is in the form of a bilinear population vector in which the lowest Fourier modes of the response vector **r** have been removed. Retaining only the high Fourier modes in the response profile suppresses the cross-correlations between the different components of the residual responses $\hat{r}_i$ because the underlying correlations have smooth spatial dependence, whose power is concentrated mostly in the low Fourier modes. On the other hand, the information contained in the variance is not removed because the variance contains a discontinuous spatial component, $C_i^d(\theta)$. In other words, the variance of a correlation profile which has only high Fourier modes can still preserve its slowly varying components. Thus, by projecting out the low Fourier modes of the spatial responses the spatial correlations are suppressed but the information in the response variance is retained.

This interpretation of the bilinear readout implies that although *all* the elements of the correlation matrix depend on the stimulus, only the stimulus dependence of the diagonal elements is important. This important conclusion is borne out by our theoretical results concerning the performance of the system. As Eqs. (11) and (19) show, the asymptotic performance of both the full FI as well as that of the OBE are equivalent to those of an uncorrelated population with a stimulus dependent variance which equals $C_i^d(\theta)$.

Although we have presented results here concerning a multiplicative model of correlations, we have studied other models of stimulus dependent correlations. These studies indicate that the above conclusions apply to a broad class of populations in which information is encoded in the second order statistics of the responses. Also, for the sake of clarity we have assumed here that the mean responses are untuned, **f** = **0**. Our studies have shown that adding tuned mean inputs does not modify the picture since the smoothly varying positive correlations greatly suppress the information embedded in the first order statistics.

The relatively simple form of the readout Eq. (22) suggests that neuronal hardware may be able to extract efficiently information embedded in local populations of cells whose noisy responses are strongly correlated, provided that the variances of their responses are significantly tuned to the stimulus. This latter condition is not too restrictive, since tuning of variances of neuronal firing rates to stimulus and motor variables is quite common in the nervous system.

**Acknowledgments**

This work was partially supported by grants from the Israel-U.S.A. Binational Science Foundation and the Israeli Science Foundation. M.S. is supported by a scholarship from the Clore Foundation.

## References

[1] E. Fetz, K. Yoyoma and W. Smith, Cerebral Cortex **9** (Plenum Press, New York, 1991).

[2] D. Lee, N.L. Port, W. Kruse and A.P. Georgopoulos, J. Neurosci. **18**, 1161 (1998).

[3] E.M. Maynard, N.G. Hatsopoulos, C.L. Ojakangas, B.D. Acuna, J.N. Sanes, R.A. Normann, and J.P. Donoghue, J. Neurosci. **19**, 8083 (1999).

[4] E. Zohary, M.N. Shadlen and W.T. Newsome, Nature **370**, 140 (1994).

[5] L.F. Abbott and P. Dayan, Neural Computation **11**, 91 (1999).

[6] H. Sompolinsky, H. Yoon, K. Kang and M. Shamir, Phys. Rev. E, **64**, 051904 (2001); H. Yoon and H. Sompolinsky, Advances in Neural Information Processing Systems 11 (pp. 167). Kearns M.J, Solla S.A and Cohn D.A, Eds., (Cambridge, MA: MIT Press, 1999).

[7] S. Seung and H. Sompolinsky, Proc. Natl. Acad. Sci. USA **90**, 10794 (1993).

[8] E. Salinas and L.F. Abbott, J. Comp. Neurosci. **1**, 89 (1994).